# Using a Saliency Map for Active Spatial Selective Attention: Implementation & Initial Results

**Shumeet Baluja**
baluja@cs.cmu.edu
School of Computer Science
Carnegie Mellon University
Pittsburgh, PA 15213

**Dean A. Pomerleau**
pomerleau@cs.cmu.edu
School of Computer Science
Carnegie Mellon University
Pittsburgh, PA 15213

## Abstract

In many vision based tasks, the ability to focus attention on the important portions of a scene is crucial for good performance on the tasks. In this paper we present a simple method of achieving spatial selective attention through the use of a saliency map. The saliency map indicates which regions of the input retina are important for performing the task. The saliency map is created through predictive auto-encoding. The performance of this method is demonstrated on two simple tasks which have multiple very strong distracting features in the input retina. Architectural extensions and application directions for this model are presented.

## 1 MOTIVATION

Many real world tasks have the property that only a small fraction of the available input is important at any particular time. On some tasks this extra input can easily be ignored. Nonetheless, often the similarity between the important input features and the irrelevant features is great enough to interfere with task performance. Two examples of this phenomena are the famous "cocktail party effect", otherwise known as speech recognition in a noisy environment, and image processing of a cluttered scene. In both cases, the extraneous information in the input signal can be easily confused with the important features, making the task much more difficult.

The concrete real world task which motivates this work is vision-based road following. In this domain, the goal is to control a robot vehicle by analyzing the scene ahead, and choosing a direction to travel based on the location of important features like lane marking and road edges. This is a difficult task, since the scene ahead is often cluttered with extraneous features such as other vehicle, pedestrians, trees, guardrails, crosswalks, road signs and many other objects that can appear on or around a roadway. [1] While we have had significant success on the road following task using simple feed-forward neural networks to transform images of the road ahead into steering commands for the vehicle [Pomerleau, 1993b], these methods fail when presented with cluttered environments like those encoun-

---

1. For the general task of autonomous navigation, these extra features are extremely important, but for restricted task of road following, which is the focus of this paper, these features are merely distractions. Although we are addressing the more general task using the techniques described here in combination with other methods, a description of these efforts is beyond the scope of this paper.

tered when driving in heavy traffic, or on city streets.

The obvious solution to this difficulty is to focus the attention of the processing system on only the relevant features by masking out the "noise". Because of the high degree of similarity between the relevant features and the noise, this filtering is often extremely difficult. Simultaneously learning to perform a task like road following and filtering out clutter in a scene is doubly difficult because of a chicken-and-egg problem. It is hard to learn which features to attend to before knowing how to perform the task, and it is hard to learn how to perform the task before knowing which features to attend to.

This paper describes a technique designed to solve this problem. It involves deriving a "saliency map" of the image from a neural network's internal representation which highlights regions of the scene considered to be important. This saliency map is used as feedback to focus the attention of the network's processing on subsequent images. This technique overcomes the chicken-and-egg problem by simultaneously learning to identify which aspects of the scene are important, and how to use them to perform a task.

## 2   THE SALIENCY MAP

A saliency map is designed to indicate which portions of the image are important for completing the required task. The trained network should be able to accomplish two goals with the presentation of each image. The first is to perform the given task using the inputs and the saliency map derived from the previous image, and the second is to predict the salient portions of the next image.

### 2.1 Implementation

The creation of the saliency map is similar to the technique of *Input Reconstruction Reliability Estimation* (IRRE) by [Pomerleau, 1993]. IRRE attempts to predict the reliability of a network's output. The prediction is made by reconstructing the input image from linear transformations of the activations in the hidden layer, and comparing it with the actual image. IRRE works on the premise that the greater the similarity between the input image and the reconstructed input image, the more the internal representation has captured the important input features, and therefore the more reliable the network's response.

A similar method to IRRE can be used to create a saliency map. The saliency map should be determined by the important features in the current image for the task to be performed. Because compressed representations of the important features in the current image are represented in the activations of the hidden units, the saliency map is derived from these, as shown in Figure 1. It should be noted that the hidden units, from which the saliency map is derived, do not necessarily contain information similar to principal components (as is achieved through auto-encoder networks), as the relevant task may only require information on a small portion of the image. In the simple architecture depicted in Figure 1, the internal representation must contain information which can be transformed by a single layer of adjustable weights into a saliency map for the next image. If such a transformation is not possible, separate hidden layers, with input from the task-specific internal representations could be employed to create the saliency map.

The saliency map is trained by using the next image, of a time-sequential data set, as the target image for the prediction, and applying standard error backpropagation on the differences between the next image and the predicted next image. The weights from the hidden

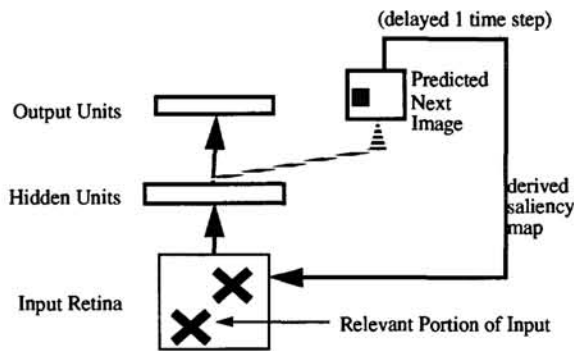

**Figure 1:** A simple architecture for using a saliency map. The dashed line represents "chilled connections", i.e. errors from these connections do not propagate back further to impact the activations of the hidden units. This architecture assumes that the target task contains information which will help determine the salient portions of the next frame.

units to the saliency map are adjusted using standard backpropagation, but the error terms are not propagated to the weights from the inputs to the hidden units. This ensures that the hidden representation developed by the network is determined only by the target task, and not by the task of prediction.

In the implementation used here, the feedback is to the input layer. The saliency map is created to either be the same size as the input layer, or is scaled to the same size, so that it can influence the representation in a straight-forward manner. The saliency map's values are scaled between 0.0 and 1.0, where 1.0 represents the areas in which the prediction matched the next image exactly. The value of 1.0 does not alter the activation of the input unit, a value of 0.0 turns off the activation. The exact construction of the saliency map is described in the next section, with the description of the experiments. The entire network is trained by standard backpropagation; in the experiments presented, no modifications to the standard training algorithm were needed to account for the feedback connections.

The training process for prediction is complicated by the potential presence of noise in the next image. The saliency map cannot "reconstruct" the noise in the next image, because it can only construct the portions of the next image which can be derived from the activation of the hidden units, which are task-specific. *Therefore, the noise in the next image will not be constructed, and thereby will be de-emphasized in the next time step by the saliency map.* The saliency map serves as a filter, which channels the processing to the important portions of the scene [Mozer, 1988]. One of the key differences between the filtering employed in this study, and that used in other focus of attention systems, is that this filtering is based on expectations from multiple frames, rather than on the retinal activations from a single frame. An alternative neural model of visual attention which was explored by [Olshausen et al., 1992] achieved focus of attention in single frames by using control neurons to dynamically modify synaptic strengths.

The saliency map may be used in two ways. It can either be used to highlight portions of the input retina or, when the hidden layer is connected in a retinal fashion using weight sharing, as in [LeCun et al., 1990], it can be used to highlight important spatial locations within the hidden layer itself. The difference is between highlighting individual pixels from which the features are developed or highlighting developed features. Discussion of the psychological evidence for both of these types of highlighting (in single-frame retinal activation based context), is given in [Pashler and Badgio, 1985].

This network architecture shares several characteristics with a Jordan-style recurrent network [Jordan, 1986], in which the output from the network is used to influence the pro-

cessing of subsequent input patterns in a temporal sequence. One important distinction is that the feedback in this architecture is spatially constrained. The saliency map represents the importance of local patches of the input, and can influence only the network's processing of corresponding regions of the input. The second distinction is that the outputs are not general task outputs, rather they are specially constructed to predict the next image. The third distinction is in the form of this influence. Instead of treating the feedback as additional input units, which contribute to the weighted sum for the network's hidden units, this architecture uses the saliency map as a gating mechanism, suppressing or emphasizing the processing of various regions of the layer to which it is fed-back. In some respects, the influence of the saliency map is comparable to the gating network in the mixture of experts architecture [Jacobs et al., 1991]. Instead of gating between the outputs of multiple expert networks, in this architecture the saliency map is used to gate the activations of the input units within the same network.

## 3  THE EXPERIMENTS

In order to study the feasibility of the saliency map without introducing other extraneous factors, we have conducted experiments with two simple tasks described below. Extensions of these ideas to larger problems are discussed in sections 4 & 5. The first experiment is representative of a typical machine vision task, in which the relevant features move very little in consecutive frames. With the method used here, the relevant features are automatically determined and tracked. However, if the relevant features were known *a priori*, a more traditional vision feature tracker which begins the search for features within the vicinity of the location of the features in the previous frame, could also perform well. The second task is one in which the feature of interest moves in a discontinuous manner. A traditional feature tracker without exact knowledge the feature's transition rules would be unable to track this feature, in the presence of the noise introduced. The transition rules of the feature of interest are learned automatically through the use of the saliency map.

In the first task, there is a slightly tilted vertical line of high activation in a 30x32 input unit grid. The width of the line is approximately 9 units, with the activation decaying with distance from the center of the line. The rest of the image does not have any activation. The task is to center a gaussian of activation around the center of the x-intercept of the line, 5 pixels above the top of the image. The output layer contains 50 units. In consecutive images, the line can have a small translational move and/or a small rotational move. Sample training examples are given in Figure 2. This task can be easily learned in the presence of no noise. The task is made much harder when lines, which have the same visual appearance as the real line (in everything except for location and tilt) randomly appear in the image. In this case, it is vital that the network is able to distinguish between the real line and noise line by using information gathered from previous image(s).

In the second task, a cross ("+") of size 5x5 appears in a 20x20 grid. There are 16 positions in which the cross can appear, as shown in Figure 2c. The locations in which the cross appears is set according to the transition rules shown in Figure 2c. The object of this problem is to reproduce the cross in a smaller 10x10 grid, with the edges of the cross extended to the edges of the grid, as shown in Figure 2b. The task is complicated by the presence of randomly appearing noise. The noise is in the form of another cross which appears exactly similar to the cross of interest. Again, in this task, it is vital for the network to be able to distinguish between the real cross, and crosses which appear as noise. As in the first task, this is only possible with knowledge of the previous image(s).

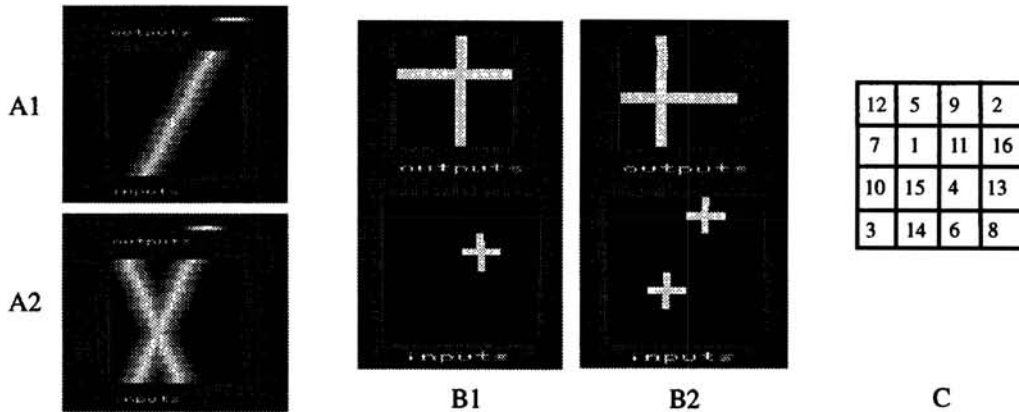

**Figure 2:** (A) The first task, image (A1) with no distractions, image (A2) with one distracting feature. (B) The second task, image (B1) with no distractions, image (B2) with two distractions. (C) Transition rules for the second task.

## 3.1 Results

The two problems described above were attempted with networks trained both with and without noise. Each of the training sessions were also tested with and without the saliency map. Each type of network was trained for the same number of pattern presentations with the same training examples. The results are shown in Table 1. The results reported in Table I represent the error accrued over 10,000 testing examples. For task 1, errors are reported in terms of the absolute difference between the peak's of the Gaussians produced in the output layer, summed for all of the testing examples (the max error per image is 49). In task 2, the errors are the sum of the absolute difference between the network's output and the target output, summed across all of the outputs and all of the testing examples.

When noise was added to the examples, it was added in the following manner (for both training and testing sets): In task 1, '1 noise' guarantees a noise element, similarly, '2 noise' guarantees two noise elements. However, in task 2, '1 noise' means that there is a 50% chance of a noise element occurring in the image, '2 noise' means that there is a 50% chance of another noise element occurring, independently of the appearance of the first noise element. The positions of the noise elements are determined randomly.

The best performance, in task 1, came from the cases in which there was no noise in testing or training, and no saliency map was used. This is expected, as this task is not difficult when no noise is present. Surprisingly, in task 2, the best case was found with the saliency map, when training *with* noise and testing without noise. This performed even better than training without noise. Investigation into this result is currently underway.

In task 1, when training and testing without noise, the saliency map can hurt performance. If the predictions made by the saliency map are not correct, the inputs appear slightly distorted; therefore, the task to be learned by the network becomes more difficult. Nevertheless, the benefit of using a saliency map is apparent when the test set contains noise.

In task 2, the network without the saliency map, trained with noise, and tested without noise cannot perform well; the performance further suffers when noise is introduced into the testing set. The noise in the training prevents accurate learning. This is not the case when the saliency map is used (Table 1, task 2). When the training set contains noise, the network with the saliency map works better when tested with and without noise.

**Table 1: Summed Error of 10,000 Testing Examples**

| Training Set | Testing Set | | | | | |
| --- | --- | --- | --- | --- | --- | --- |
| | Task 1 | | | Task 2 | | |
| | 0 Noise | 1 Noise | 2 Noise | 0 Noise | 1 Noise | 2 Noise |
| 0 Noise (Saliency) | 12672 | 60926 | 82282 | 7174 | 94333 | 178883 |
| 0 Noise (No Saliency) | 10241 | 91812 | 103605 | 7104 | 133496 | 216884 |
| 1 Noise (Saliency) | 18696 | 26178 | 52668 | 4843 | 10427 | 94422 |
| 1 Noise (No Saliency) | 14336 | 80668 | 97433 | 31673 | 150078 | 227650 |

When the noise increased beyond the level of training, to 2 noise elements per image, the performances of networks trained both with and without the saliency map declined. It is suspected that training the networks with increased noise will improve performance in the network trained with the saliency map. Nonetheless, due to the amount of noise compared to the small size of the input layer, improvements in results may not be dramatic.

In Figure 3, a typical test run of the second task is shown. In the figure, the inputs, the predicted and actual outputs, and the predicted and actual saliency maps, are shown. The actual saliency map is just a smaller version of the unfiltered next input image. The input size is 20x20, the outputs are 10x10, and the saliency map is 10x10. The saliency map is scaled to 20x20 when it is applied to the next inputs. Note that in the inputs to the network, one cross appears much brighter than the other; this is due to the suppression of the distracting cross by the saliency map. The particular use of the saliency map which is employed in this study, proceeded as follows: the difference between the saliency map (derived from input image$_i$) and the input image$_{i+1}$ was calculated. This difference image was scaled to the range of 0.0 to 1.0. Each pixel was then passed through a sigmoid; alternatively, a hard-step function could have been used. This is the saliency map. The saliency map was multiplied by input image$_{i+1}$; this was used as the input into the network. If the sigmoid is not used, features, such as incomplete crosses, sometimes appear in the input image. This happens because different portions of the cross may have slightly different saliency values associated with them -due to errors in prediction coupled with the scaling of the saliency map. Although the sigmoid helps to alleviate the problem, it does not eliminate it. This explains why training with no noise with a saliency map sometimes does not perform as well as training without a saliency map.

## 4 ALTERNATIVE IMPLEMENTATIONS

An alternative method of implementing the saliency map is with standard additive connections. However, these connection types have several drawbacks in comparison with the multiplicative ones use in this study. First, the additive connections can drastically change the meaning of the hidden unit's activation by changing the sign of the activation. The saliency map is designed to indicate which regions are important for accomplishing the task based upon the features in the hidden representations; as little alteration of the important features as possible is desired. Second, if the saliency map is incorrect, and suggests an area of which is not actually important, the additive connections will cause 'ghost' images to appear. These are activations which are caused *only* by the influence of the addi-

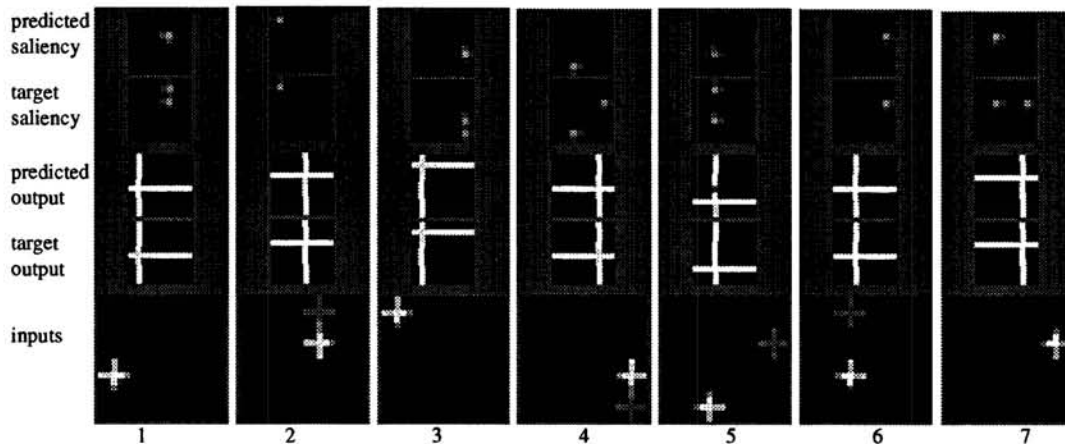

predicted
saliency

target
saliency

predicted
output

target
output

inputs

1    2    3    4    5    6    7

**Figure 3:** A typical sequence of inputs and outputs in the second task. Note that when two crosses are in the inputs, one is much brighter than the other. The "noise" cross is de-emphasized.

tive saliency map. The multiplicative saliency map, as is implemented here, does not have either of these problems.

A second alternative, which is more closely related to a standard recurrent network [Jordan, 1986], is to use the saliency map as extra inputs into the network. The extra inputs serve to indicate the regions which are expected to be important. Rather than hand-coding the method to represent the importance of the regions to the network, as was done in this study, the network learns to use the extra inputs when necessary. Further, the saliency map serves as the predicted next input. This is especially useful when the features of interest may have momentarily been partially obscured or have entirely disappeared from the image. This implementation is currently being explored by the authors for use in a autonomous road lane-tracking system in which the lane markers are not always present in the input image.

## 5  CONCLUSIONS & FUTURE DIRECTIONS

These experiments have demonstrated that an artificial neural network can learn to both identify the portions of a visual scene which are important, and to use these important features to perform a task. The selective attention architecture we have develop uses two simple mechanisms, predictive auto-encoding to form a saliency map, and a constrained form of feedback to allow this saliency map to focus processing in the subsequent image.

There are at least four broad directions in which this research should be extended. The first is, as described here, related to using the saliency map as a method for automatically actively focusing attention to important portions of the scene. Because of the spatial dependence of the task described in this paper, with the appropriate transformations, the output units could be directly fed back to the input layer to indicate saliency. Although this does not weaken the result, in terms of the benefit of using a saliency map, future work should also focus on problems which do not have this property to determine how easily a saliency map can be constructed. Will the use of weight sharing be enough to develop the necessary spatially oriented feature detectors? Harder problems are those with target tasks which does not explicitly contain spatial saliency information.

An implementation problem which needs to be resolved is in networks which contain more than a single hidden layer: from which layer should the saliency map be con-

structed? The trade-off is that at the higher layers, the information contained is more task specific. However, the higher layers may effectively extract the information required to perform the task, without maintaining the information required for saliency map creation. The opposite case is true in the lower layers; these may contain all of the information required, but may not provide enough discrimination to narrow the focus effectively.

The third area for research is an alternative use for the saliency map. ANNs have often been criticized for their uninterpretability, and lack of mechanism to explain performance. The saliency map provides a method for understanding, at a high level, what portions of the inputs the ANN finds the most important.

Finally, the fourth direction for research is the incorporation of higher level, or symbolic knowledge. The saliency map provides a very intuitive and direct method for focusing the network's attention to specific portions of the image. The saliency map may prove to be a useful mechanism to allow other processes, including human users, to simply "point at" the portion of the image to which the network should be paying attention.

The next step in our research is to test the effectiveness of this technique on the main task of interest, autonomous road following. Fortunately, the first demonstration task employed in this paper shares several characteristics with road following. Both tasks require the network to track features which move over time in a cluttered image. Both tasks also require the network to produce an output that depends on the positions of the important features in the image. Because of these shared characteristics, we believe that similar performance improvements should be possible in the autonomous driving domain.

## Acknowledgments

Shumeet Baluja is supported by a National Science Foundation Graduate Fellowship. Support has come from "Perception for Outdoor Navigation" (contract number DACA76-89-C-0014, monitored by the US Army Topographic Engineering Center) and "Unmanned Ground Vehicle System" (contract number DAAE07-90-C-R059, monitored by TACOM). Support has also come from the National Highway Traffic Safety Administration under contract number DTNH22-93-C-07023. The views and conclusions contained in this document are those of the authors and should not be interpreted as representing official policies, either expressed or implied, of the National Science Foundation, ARPA, or the U.S. Government.

## References

Cottrell, G.W. & Munro, P. (1988) Principal Component Analysis of Images via back-propagation. *Proc Soc. of Photo-Optical Instr. Eng.*, Cambridge, MA.

Jordan, M.I., (1989). Serial Order: A Parallel, Distributed Processing Approach. In *Advances in Connectionist Theory: Speech*, eds. J.L. Elman and D.E. Rumerlhart. Hillsdale: Erlbaum.

Jacobs, R.A., Jordan, M.I., Nowlan, S.J. & Hinton, G.E. (1991). Adaptive Mixtures of Local Experts. *Neural Computation*, 3:1.

LeCun, Y., Boser, B., Denker, J.S., Henderson, D. Howard, R.E., Hummand W., and Jackel, L.D. (1989) Back-propagation Applied to Handwritten Zip Code Recognition. Neural Computation 1, 541-551. MIT, 1989.

Mozer, M.C. (1988) A Connectionist Model of Selective Attention in Visual Perception. Technical Report, University of Toronto, CRG-TR-88-4.

Pashler, H. & Badgio, P. (1985). Visual Attention and Stimulus Identification. Journal of Experimental Psychology: Human Perception and Performance, 11 105-121.

Pomerleau, D.A. (1993) Input Reconstruction Reliability Estimation. In Giles, C.L. Hanson, S.J. and Cowan, J.D. (eds). *Advances in Neural Information Processing Systems 5*, CA: Morgan Kaufmann Publishers.

Pomerleau, D.A. (1993b) *Neural Network Perception for Mobile Robot Guidance*, Kluwer Academic Publishing.

Olshausen, B., Anderson, C., & Van Essen; D. (1992) A Neural Model of Visual Attention and Invariant Pattern Recognition. California Institute of Technology, CNS Program, memo-18.